# Reconfigurable Neural Net Chip with 32K Connections

**H.P. Graf, R. Janow, D. Henderson, and R. Lee**
AT&T Bell Laboratories, Room 4G320, Holmdel, NJ 07733

## Abstract

We describe a CMOS neural net chip with a reconfigurable network architecture. It contains 32,768 binary, programmable connections arranged in 256 'building block' neurons. Several 'building blocks' can be connected to form long neurons with up to 1024 binary connections or to form neurons with analog connections. Single- or multi-layer networks can be implemented with this chip. We have integrated this chip into a board system together with a digital signal processor and fast memory. This system is currently in use for image processing applications in which the chip extracts features such as edges and corners from binary and gray-level images.

## 1 INTRODUCTION

A key problem for a hardware implementation of neural nets is to find the proper network architecture. With a fixed network structure only few problems can be solved efficiently. Therefore, we opted for a programmable architecture that can be changed under software control. A large, fully interconnected network can, in principle, implement any architecture, but this usually wastes a lot of the connections since many have to be set to zero. To make better use of the silicon, other designs implemented a programmable architecture - either by connecting several chips with switching blocks (Mueller89), or by placing switches between blocks of synapses on one chip (Satyanarayana90). The present design (Graf90) consists of building blocks that can be connected to form many different network configurations. Single-layer

nets or multi-layer nets can be implemented. The connections can be binary or can have an analog depth of up to four bits.

We designed this neural net chip mainly for pattern recognition applications, which typically require nets far too large for a single chip. However, the nets can often be structured so that the neurons have local receptive fields, and many neurons share the same receptive field. Such nets can be split into smaller parts that fit onto a chip, and the small nets are then scanned sequentially over an image. The circuit has been optimized for this type of network by adding shift registers for the data transport.

The neural net chip implementation uses a mixture of analog and digital electronics. The weights, the neuron states and all the control signals are digital, while summing the contributions of all the weights is performed in analog form. All the data going on and off the chip are digital, which makes the integration of the network into a digital system straight-forward.

## 2    THE CIRCUIT ARCHITECTURE

### 2.1    The Building Block

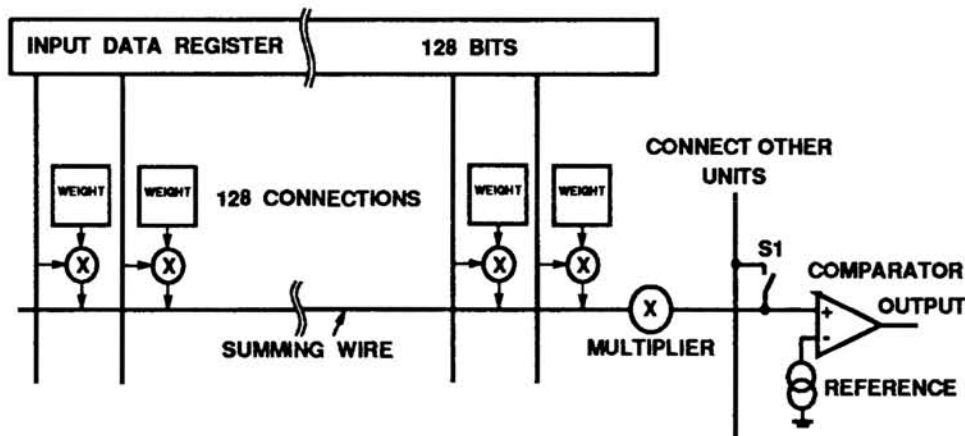

Figure 1: One of the building blocks, a "neuron"

Figure 1 shows schematically one of the building blocks. It consists of an array of 128 connections which receive input signals from other neurons or from external sources. The weights as well as the inputs have binary values, +1 or -1. The output of a connection is a current representing the result of the multiplication of a weight with a state, and on a wire the currents from all the connections are summed. This sum is multiplied with a programmable factor and can be added to the currents of other neurons. The result is compared with a reference and is thresholded in a comparator. A total of 256 such building blocks are contained on the chip.

Up to 8 of the building blocks can be connected to form a single neuron with up to 1024 connections. The network is not restricted to binary connections. Connections with four bits of analog depth are obtained by joining four building blocks and by setting each of the multipliers to a different value: 1, 1/2, 1/4, 1/8 (see Figure

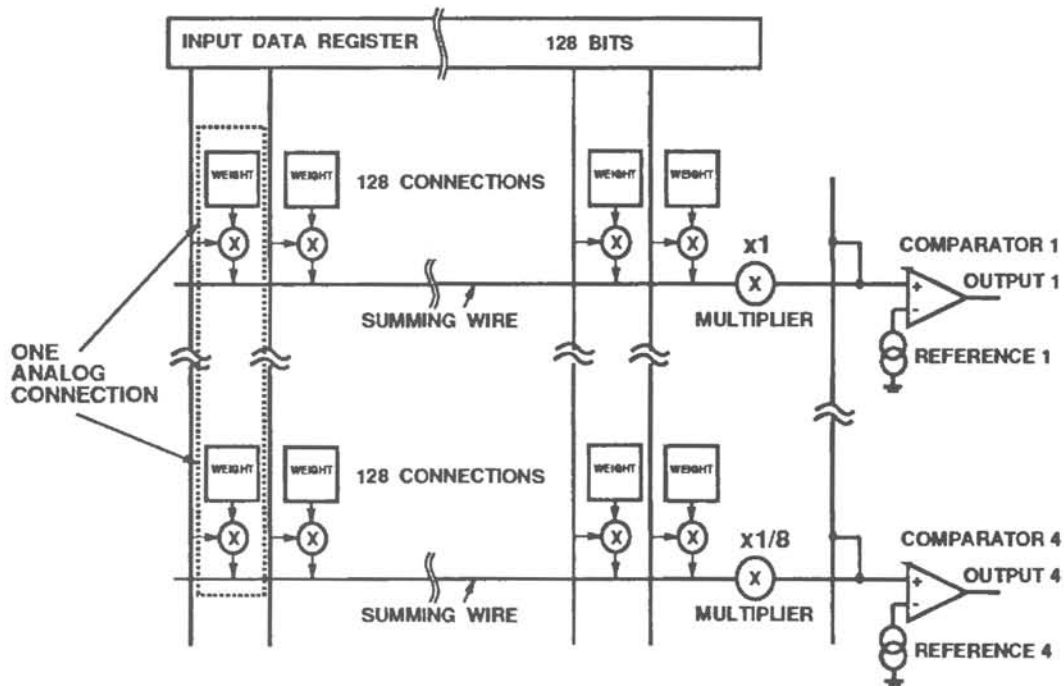

Figure 2: Connecting four building blocks to form connections with four bits of resolution

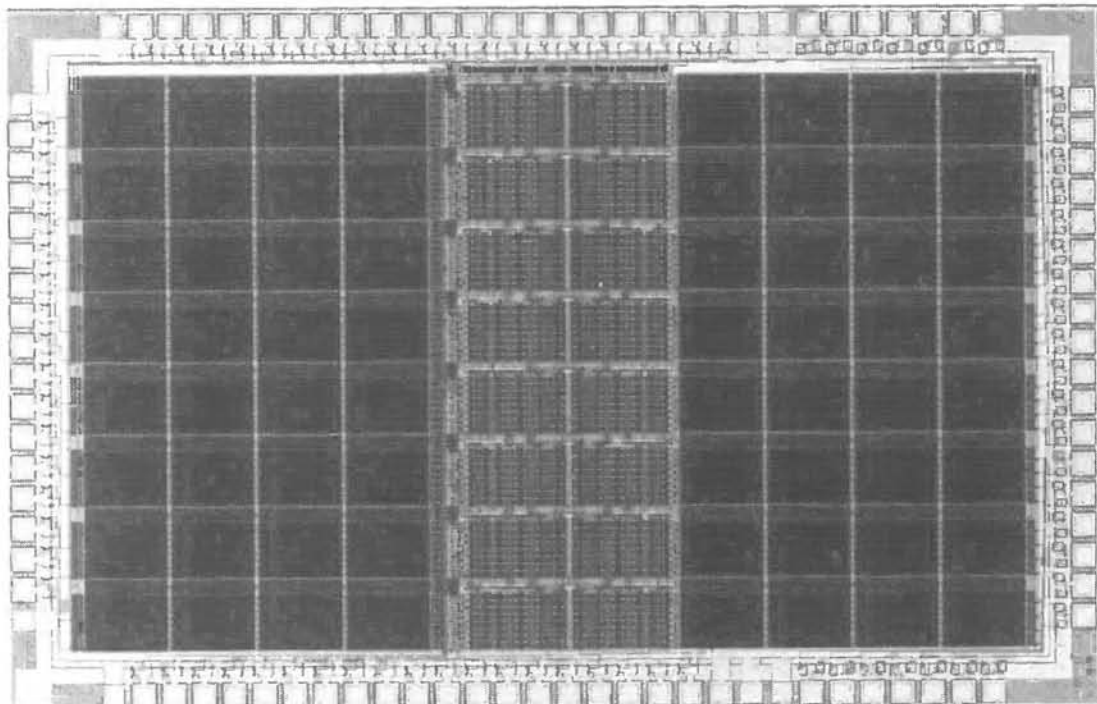

Figure 3: Photo micrograph of the neural net chip

2). In this case four binary connections, one in each building block, form one connection with an analog depth of four bits. Alternatively, the network can be configured for two-bit input signals and two-bit weights or four-bit inputs and one-bit weights. The multiplications of the input signals with the weights are four-quadrant multiplications, whether binary signals are used or multi-bit signals. With this approach only one scaling multiplier is needed per neuron, instead of one per connection as would be the case if connections were implemented with multiplying D/A converters.

The transfer function of a neuron is provided by the comparator. With a single comparator the transfer function has a hard threshold. Other types of transfer functions can be obtained when several building blocks are connected. Then, several comparators receive the same analog input and for each comparator a different reference can be selected (compare figure 2). In this way, for example, eight comparators may work as a three-bit A/D converter. Other transfer functions, such as sigmoids, can be approximated by selecting appropriate thresholds for the comparators.

The neurons are arranged in groups of 16. For each group there is one register of 128 bits providing the input data. The whole network contains 16 such groups, split in two halfs, each with eight groups. These groups of neurons can be recognized in Figure 3 that shows a photomicrograph of the circuit. The chip contains 412,000 transistors and measures 4.5mm x 7mm. It is fabricated in $0.9\mu$m CMOS technology with one level of poly and two levels of metal.

## 2.2   Moving Data Through The Circuit

From a user's point of view the chip consists of the four different types of registers listed in table 1. Programming of the chip consists in moving the data over a high-speed bus of 128 bits width between these registers. Results produced by the network can be loaded directly into data-input registers and can be used for a next computation. In this way some multi-layer networks can be implemented without loading data off chip between layers.

Table 1: Registers in the neural net chip

| REGISTER | FUNCTION |
| --- | --- |
| Shift register | Input and output of the data |
| Data-input registers | Provide input to the connections |
| Configuration registers | Determine the connectivity of the network |
| Result registers | Contain the output of the network |

In a typical operation 16 bits are loaded from the outside into a shift register. From that register the main bus distributes the data through the whole circuit. They are loaded into one or several of the data-input registers. The analog computation is then started and the results are latched into the result registers. These results are loaded either into data-input registers if a network with feedback or a multi-layer network is implemented, or they are loaded into the output shift register and off chip.

In addition to the main bus, the chip contains two 128 bit wide shift registers, one through each half of the connection matrix. All the shift registers were added to speed up the operation when networks with local fields of view are scanned over a signal. In such an application shift registers drastically reduce the amount of new data that have to be loaded into the chip from one run to the next. For example, when an input field of 16 x 16 pixels is scanned over an image, only 16 new data values have to be loaded for each run instead of 256. Loading the data on and off the chip is often the speed-limiting operation. Therefore, it is important to provide some support in hardware.

## 3    TEST RESULTS

The speed of the circuit is limited by the time it takes the analog computation to settle to its final value. This operation requires less than 50 ns. The chip can be operated with instruction cycles of 100ns, where in the first 50 ns the analog computation settles down and during the following 50 ns the results are read out. Simultaneously with reading out the results, new data are loaded into the data-input registers. In this way 32k one-bit multiply-accumulates are executed every 100ns, which amounts to 320 billion connections/second.

The accuracy of the analog computation is about ±5%. This means, for example, that a comparator whose threshold is set to a value of 100 connections may already turn on when it receives the current from 95 connections. This limited accuracy is due to mismatches of the devices used for the analog computation. However, the threshold for each comparator may be individually adjusted at the cost of dedicating neurons to the task. Then a threshold can be adjusted to ±1%. The operation of the digital part of the network and the analog part has been synchronized in such a way that the noise generated by the digital part has a minimal effect on the analog computation.

## 4    THE BOARD SYSTEM

A system was developed to use the neural net chip as a coprocessor of a workstation with a VME bus. A schematic of this system is shown in figure 4. Beside the neural net chip, the board contains a digital signal processor to control the whole system and 256k of fast memory. Pictures are loaded from the host into the board's memory and are then scanned with the neural net chip. The results are loaded back into the board memory and from there to the host.

Loading pictures over the VME bus limits the overall speed of this system to about one frame of 512 x 512 pixels per second. Although this corresponds to less than 10% of the chips maximum data throughput, operations such as scanning an image with 32 16 x 16 kernels can be done in one second. The same operation would take around 30 minutes on the workstation alone. Therefore, this system represents a very useful tool for image processing, in particular for developing algorithms. Its architecture makes it very flexible since part of a problem can be solved by the digital signal processor and the computationally intensive parts on the neural net chip. An extra data path for the signals will be added later to take full advantage of the neural net's speed.

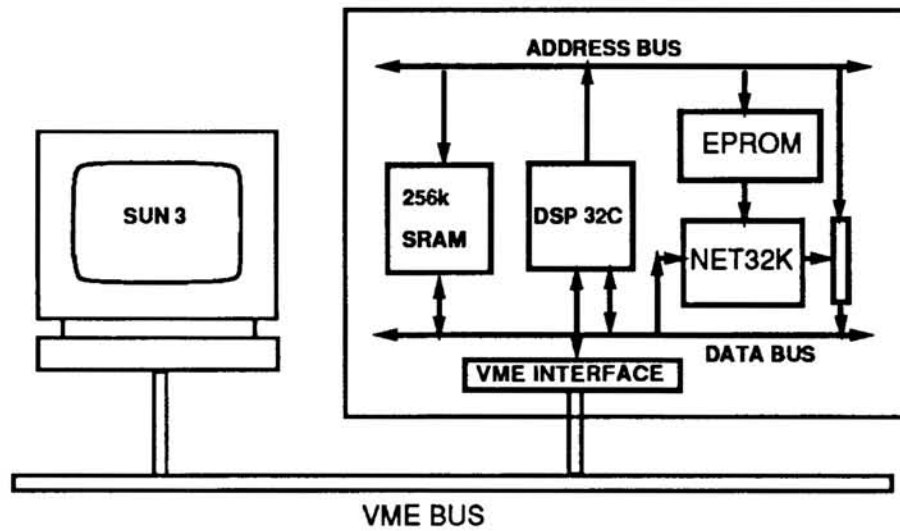

Figure 4: Schematic of the board system for the neural net chip

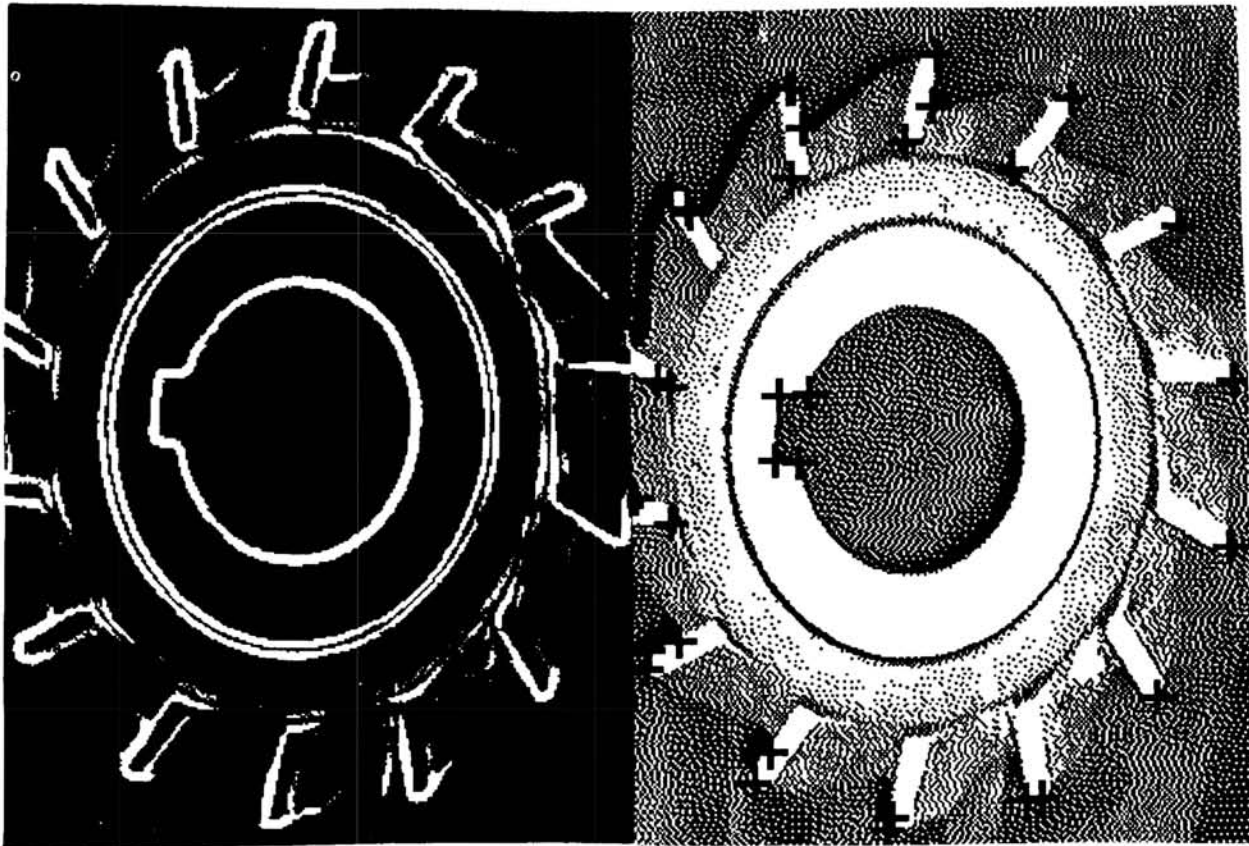

Figure 5: Result of a feature extraction application. Left image: Edges extracted from the milling cutter. Right image: The crosses mark where corners were detected. A total of 16 features were extracted simultaneously with detectors of 16 x 16 pixels in size.

## 5  APPLICATIONS

Figure 5 shows the result of an application, where the net is used for extracting simultaneously edges and corners from a gray-level image. The network actually handles only a small resolution in the pixel values. Therefore, the picture is first half-toned with a standard error-diffusion algorithm and then, the halftoned image is scanned with the network. To extract these features, kernels with three levels in the weights are loaded into the network. One neuron with 256 two-bit connections represents one kernel. There are a total of 16 such kernels in the network, each one tuned to a corner or an edge of a different orientation. For each comparator an extra neuron is used to set the threshold. This whole task fills 50% of the chip.

Edges and corners are important features that are often used to identify objects or to determine their positions and orientations. We are applying them now to segment complex images. Convolutional algorithms have long been recognized as reliable methods for extracting features. However, they are computationally very expensive so that often special-purpose hardware is required. To our knowledge, no other circuit can extract such a large number of features at a comparable rate.

This application demonstrates, how a large number of connections can compensate for a limited resolution in the weights and the states. We took a gray level image and clipped its pixels to binary values. Despite this coarse quantization of the signal the relevant information can be extracted reliably. Since many connections are contributing to each result, uncorrelated errors due to quantization are averaged out. The key to a good result is to make sure that the quantization errors are indeed uncorrelated, at least approximately.

This circuit has been designed with pattern matching applications in mind. However, its flexibility makes it suitable for a much wider range of applications. In particular, since its connections as well as its architecture can be changed fast, in the order of 100ns, it can be integrated in an adaptive or a learning system.

### Acknowledgements

We acknowledge many stimulating discussions with the other members of the neural network group at AT&T in Holmdel. Part of this work was supported by the USASDC under contract #DASG60-88-0044.

### References

H. P. Graf & D. Henderson. (1990) A Reconfigurable CMOS Neural Network. in *Digest IEEE Int. Solid State Circuits Conf.* , 144-145.

P. Mueller, J. van der Spiegel, D. Blackman, T. Chiu, T. Clare, J. Dao, Ch. Donham, T.P. Hsieh & M. Loinaz. (1989) A Programmable Analog Neural Computer and Simulator. In D.S. Touretzky (ed.), *Advances in Neural Information Processing Systems 1*, 712 - 719. San Mateo, CA: Morgan Kaufmann.

S. Satyanarayana, Y. Tsividis & H. P. Graf. (1990) A Reconfigurable Analog VLSI Neural Network Chip. In D.S. Touretzky (ed.), *Advances in Neural Information Processing Systems 2*, 758 - 768. San Mateo, CA: Morgan Kaufmann.